# CPRL – An Extension of Compressive Sensing to the Phase Retrieval Problem

**Henrik Ohlsson**

Division of Automatic Control, Department of Electrical Engineering,
Linköping University, Sweden.
Department of Electrical Engineering and Computer Sciences
University of California at Berkeley, CA, USA
ohlsson@eecs.berkeley.edu

**Allen Y. Yang**

Department of Electrical Engineering and Computer Sciences
University of California at Berkeley, CA, USA

**Roy Dong**

Department of Electrical Engineering and Computer Sciences
University of California at Berkeley, CA, USA

**S. Shankar Sastry**

Department of Electrical Engineering and Computer Sciences
University of California at Berkeley, CA, USA

## Abstract

While compressive sensing (CS) has been one of the most vibrant research fields in the past few years, most development only applies to linear models. This limits its application in many areas where CS could make a difference. This paper presents a novel extension of CS to the phase retrieval problem, where intensity measurements of a linear system are used to recover a complex sparse signal. We propose a novel solution using a lifting technique – CPRL, which relaxes the NP-hard problem to a nonsmooth semidefinite program. Our analysis shows that CPRL inherits many desirable properties from CS, such as guarantees for exact recovery. We further provide scalable numerical solvers to accelerate its implementation.

## 1  Introduction

In the area of X-ray imaging, *phase retrieval* (PR) refers to the problem of recovering a complex multivariate signal from the squared magnitude of its Fourier transform. Existing sensor devices for collecting X-ray images are only sensitive to signal intensities but not the phases. However, it is very important to be able to recover the missing phase information as it reveals finer structures of the subjects than using the intensities alone. The PR problem also has broader applications and has been studied extensively in biology, physics, chemistry, astronomy, and more recent nanosciences [29, 20, 18, 24, 23].

Mathematically, PR can be formulated using a linear system $\boldsymbol{y} = A\boldsymbol{x} \in \mathbb{C}^N$, where the matrix $A$ may represent the Fourier transform or other more general linear transforms. If the complex measurements $\boldsymbol{y}$ are available and the matrix $A$ is assumed given, it is well known that the least-squares (LS) solution recovers the model parameter $\boldsymbol{x}$ that minimizes the squared estimation error:

$\|\boldsymbol{y} - A\boldsymbol{x}\|_2^2$. In PR, we assume that the phase of the coefficients of $\boldsymbol{y}$ is omitted and only the squared magnitude of the output is observed:

$$b_i = |y_i|^2 = |\langle \boldsymbol{x}, \boldsymbol{a}_i \rangle|^2, \quad i = 1, \cdots, N, \tag{1}$$

where $A^H = [\boldsymbol{a}_1, \cdots, \boldsymbol{a}_N] \in \mathbb{C}^{n \times N}$, $\boldsymbol{y}^T = [y_1, \cdots, y_N] \in \mathbb{C}^N$, and $A^H$ denotes the Hermitian transpose of $A$.

Inspired by the emerging theory of compressive sensing [17, 8] and a lifting technique recently proposed for PR [13, 10], we study the PR problem with a more restricted assumption that the model parameter $\boldsymbol{x}$ is sparse and the number of observations $N$ are too few for (1) to have a unique solution, and in some cases even *fewer* measurements than the number of unknowns $n$. The problem is known as *compressive phase retrieval* (CPR) [25, 27, 28]. In many X-ray imaging applications, for instance, if the complex source signal is indeed sparse under a proper basis, CPR provides a viable solution to exactly recover the signal while collecting much fewer measurements than the traditional non-compressive solutions.

Clearly, the PR problem and its CPR extension are much more challenging than the LS problem, as the phase of $\boldsymbol{y}$ is lost while only its squared magnitude is available. For starters, it is important to note that the setup naturally leads to ambiguous solutions regardless whether the original linear model is overdetermined or not. For example, if $\boldsymbol{x}_0 \in \mathbb{C}^n$ is a solution to $\boldsymbol{y} = A\boldsymbol{x}$, then any multiplication of $\boldsymbol{x}$ and a scalar $c \in \mathbb{C}$, $|c| = 1$, leads to the same squared output $\boldsymbol{b}$. As mentioned in [10], when the dictionary $A$ represents the unitary discrete Fourier transform (DFT), the ambiguities may represent time-reversed or time-shifted solutions of the ground-truth signal. Hence, these global ambiguities are considered acceptable in PR applications. In this paper, when we talk about a *unique* solution to PR, it is indeed a representative of a family of solutions up to a global phase ambiguity.

## 1.1 Contributions

The main contribution of the paper is a convex formulation of the CPR problem. Using the lifting technique, the NP-hard problem is relaxed as a *semidefinite program* (SDP). We will briefly summarize several theoretical bounds for guaranteed recovery of the complex input signal, which is presented in full detail in our technical report [26]. Built on the assurance of the guaranteed recovery, we will focus on the development of a novel scalable implementation of CPR based on the *alternating direction method of multipliers* (ADMM) approach. The ADMM implementation provides a means to apply CS ideas to PR applications *e.g.,* high-impact nanoscale X-ray imaging.

In the experiment, we will present a comprehensive comparison of the new algorithm with the traditional interior-point method, other state-of-the-art sparse optimization techniques, and a greedy algorithm proposed in [26]. In high-dimensional complex domain, the ADMM algorithm demonstrates superior performance in our simulated examples and real images. Finally, the paper also provides practical guidelines to practitioners at large working on other similar nonsmooth SDP applications. To aid peer evaluation, the source code of all the algorithms have been made available at: http://www.rt.isy.liu.se/~ohlsson/.

## 2 Compressive Phase Retrieval via Lifting (CPRL)

Since (1) is nonlinear in the unknown $\boldsymbol{x}$, $N \gg n$ measurements are in general needed for a unique solution. When the number of measurements $N$ are fewer than necessary for such a unique solution, additional assumptions are needed as regularization to select one of the solutions. In classical CS, the ability to find the sparsest solution to a linear equation system enables reconstruction of signals from far fewer measurements than previously thought possible. Classical CS is however only applicable to systems with linear relations between measurements and unknowns. To extend classical CS to the nonlinear PR problem, we seek the sparsest solution satisfying (1):

$$\min_{\boldsymbol{x}} \|\boldsymbol{x}\|_0, \quad \text{subj. to} \quad \boldsymbol{b} = |A\boldsymbol{x}|^2 = \{\boldsymbol{a}_i^H \boldsymbol{x}\boldsymbol{x}^H \boldsymbol{a}_i\}_{1 \le i \le N}, \tag{2}$$

with the square acting element-wise and $\boldsymbol{b} = [b_1, \cdots, b_N]^T \in \mathbb{R}^N$. As the counting norm $\| \cdot \|_0$ is not a convex function, following the $\ell_1$-norm relaxation in CS, (2) can be relaxed as

$$\min_{\boldsymbol{x}} \|\boldsymbol{x}\|_1, \quad \text{subj. to} \quad \boldsymbol{b} = |A\boldsymbol{x}|^2 = \{\boldsymbol{a}_i^H \boldsymbol{x}\boldsymbol{x}^H \boldsymbol{a}_i\}_{1 \le i \le N}. \tag{3}$$

Note that (3) is still not a convex program, as its equality constraint is not a linear equation. In the literature, a lifting technique has been extensively used to reframe problems such as (3) to a standard form in SDP, such as in Sparse PCA [15]. More specifically, given the ground-truth signal $\boldsymbol{x}_0 \in \mathbb{C}^n$, let $X_0 \triangleq \boldsymbol{x}_0 \boldsymbol{x}_0^H \in \mathbb{C}^{n \times n}$ be an induced rank-1 semidefinite matrix. Then (3) can be reformulated into[1]

$$\min_{X \succeq 0} \|X\|_1, \quad \text{subj. to} \quad \text{rank}(X) = 1, \ b_i = \boldsymbol{a}_i^H X \boldsymbol{a}_i, \ i = 1, \cdots, N. \tag{4}$$

This is of course still a nonconvex problem due to the rank constraint. The lifting approach addresses this issue by replacing $\text{rank}(X)$ with $\text{Tr}(X)$. For a positive-semidefinite matrix, $\text{Tr}(X)$ is equal to the sum of the eigenvalues of $X$ (or the $\ell_1$-norm on a vector containing all eigenvalues of $X$). This leads to the nonsmooth SDP

$$\min_{X \succeq 0} \text{Tr}(X) + \lambda \|X\|_1, \quad \text{subj. to} \quad b_i = \text{Tr}(\Phi_i X), \ i = 1, \cdots, N, \tag{5}$$

where we further denote $\Phi_i \triangleq \boldsymbol{a}_i \boldsymbol{a}_i^H \in \mathbb{C}^{n \times n}$ and $\lambda \geq 0$ is a design parameter. Finally, the estimate of $\boldsymbol{x}$ can be found by computing the rank-1 decomposition of $X$ via singular value decomposition. We refer to the approach as *compressive phase retrieval via lifting* (CPRL).

Consider now the case that the measurements are contaminated by data noise. In a linear model, bounded random noise typically affects the output of the system as $\boldsymbol{y} = A\boldsymbol{x} + \boldsymbol{e}$, where $\boldsymbol{e} \in \mathbb{C}^N$ is a noise term with bounded $\ell_2$-norm: $\|\boldsymbol{e}\|_2 \leq \epsilon$. However, in phase retrieval, we follow closely a more special noise model used in [13]:

$$b_i = |\langle \boldsymbol{x}, \boldsymbol{a}_i \rangle|^2 + e_i. \tag{6}$$

This nonstandard model avoids the need to calculate the squared magnitude output $|\boldsymbol{y}|^2$ with the added noise term. More importantly, in most practical phase retrieval applications, measurement noise is introduced when the squared magnitudes or intensities of the linear system are measured on the sensing device, but not $\boldsymbol{y}$ itself. Accordingly, we denote a linear operator $B$ of $X$ as

$$B : X \in \mathbb{C}^{n \times n} \mapsto \{\text{Tr}(\Phi_i X)\}_{1 \leq i \leq N} \in \mathbb{R}^N, \tag{7}$$

which measures the noise-free squared output. Then the approximate CPR problem with bounded $\ell_2$-norm error model can be solved by the following nonsmooth SDP program:

$$\min_{X \succeq 0} \text{Tr}(X) + \lambda \|X\|_1, \quad \text{subj. to} \quad \|B(X) - \boldsymbol{b}\|_2 \leq \varepsilon. \tag{8}$$

Due to the machine rounding error, in general a nonzero $\varepsilon$ should be always assumed and in its termination condition during the optimization. The estimate of $\boldsymbol{x}$, just as in noise free case, can finally be found by computing the rank-1 decomposition of $X$ via singular value decomposition. We refer to the method as *approximate CPRL*.

## 3 Theoretical Analysis

This section highlights some of the analysis results derived for CPRL. The proofs of these results are available in the technical report [26]. The analysis follows that of CS and is inspired by derivations given in [13, 12, 16, 9, 3, 7]. In order to state some theoretical properties for CPRL, we need a generalization of the restricted isometry property (RIP).

**Definition 1 (RIP)** *A linear operator $B(\cdot)$ as defined in (7) is $(\epsilon, k)$-RIP if $\left| \frac{\|B(X)\|_2^2}{\|X\|_2^2} - 1 \right| < \epsilon$ for all $\|X\|_0 \leq k$ and $X \neq 0$.*

We can now state the following theorem:

**Theorem 2 (Recoverability/Uniqueness)** *Let $B(\cdot)$ be a $(\epsilon, 2\|X^*\|_0)$-RIP linear operator with $\epsilon < 1$ and let $\bar{\boldsymbol{x}}$ be the sparsest solution to (1). If $X^*$ satisfies $\boldsymbol{b} = B(X^*)$, $X^* \succeq 0$, $\text{rank}\{X^*\} = 1$, then $X^*$ is unique and $X^* = \bar{\boldsymbol{x}} \bar{\boldsymbol{x}}^H$.*

We can also give a bound on the sparsity of $\bar{\boldsymbol{x}}$:

**Theorem 3 (Bound on $\|\bar{\boldsymbol{x}} \bar{\boldsymbol{x}}^H\|_0$ from above)** *Let $\bar{\boldsymbol{x}}$ be the sparsest solution to (1) and let $\tilde{X}$ be the solution of CPRL (5). If $\tilde{X}$ has rank 1 then $\|\tilde{X}\|_0 \geq \|\bar{\boldsymbol{x}} \bar{\boldsymbol{x}}^H\|_0$.*

The following result now holds trivially:

**Corollary 4 (Guaranteed recovery using RIP)** *Let $\bar{x}$ be the sparsest solution to* (1). *The solution of CPRL $\tilde{X}$ is equal to $\bar{x}\bar{x}^H$ if it has rank 1 and $B(\cdot)$ is $(\epsilon, 2\|\tilde{X}\|_0)$-RIP with $\epsilon < 1$.*

If $\bar{x}\bar{x}^H = \tilde{X}$ can not be guaranteed, the following bound becomes useful:

**Theorem 5 (Bound on $\|X^* - \tilde{X}\|_1$)** *Let $\epsilon < \frac{1}{1+\sqrt{2}}$ and assume $B(\cdot)$ to be a $(\epsilon, 2k)$-RIP linear operator. Let $X^*$ be any matrix (sparse or dense) satisfying $b = B(X^*)$, $X^* \succeq 0$, $\mathrm{rank}\{X^*\} = 1$, let $\tilde{X}$ be the CPRL solution,* (5), *and form $X_s$ from $X^*$ by setting all but the $k$ largest elements to zero. Then,*

$$(1 - (\tfrac{2\sqrt{k}}{1-\rho}+1)\tfrac{1}{\lambda})\|\tilde{X} - X^*\|_1 \leq \tfrac{2}{(1-\rho)\sqrt{k}}\|X^* - X_s\|_1, \tag{9}$$

*with $\rho = \sqrt{2}\epsilon/(1-\epsilon)$.*

Given the RIP analysis, it may be the case that the linear operator $B(\cdot)$ does not well satisfy the RIP property defined in Definition 1, as pointed out in [13]. In these cases, RIP-1 maybe considered:

**Definition 6 (RIP-1)** *A linear operator $B(\cdot)$ is $(\epsilon, k)$-RIP-1 if $\left|\frac{\|B(X)\|_1}{\|X\|_1} - 1\right| < \epsilon$ for all matrices $X \neq 0$ and $\|X\|_0 \leq k$.*

Theorems 2–3 and Corollary 4 all hold with RIP replaced by RIP-1 and are not restated in detail here. Instead we summarize the most important property in the following theorem:

**Theorem 7 (Upper bound & recoverability through $\ell_1$)** *Let $\bar{x}$ be the sparsest solution to* (1). *The solution of CPRL* (5), *$\tilde{X}$, is equal to $\bar{x}\bar{x}^H$ if it has rank 1 and $B(\cdot)$ is $(\epsilon, 2\|\tilde{X}\|_0)$-RIP-1 with $\epsilon < 1$.*

The RIP type of argument may be difficult to check for a given matrix and are more useful for claiming results for classes of matrices/linear operators. For instance, it has been shown that random Gaussian matrices satisfy the RIP with high probability. However, given realization of a random Gaussian matrix, it is indeed difficult to check if it actually satisfies the RIP. Two alternative arguments are spark [14] and mutual coherence [17, 11]. The spark condition usually gives tighter bounds but is known to be difficult to compute as well. On the other hand, mutual coherence may give less tight bounds, but is more tractable. We will focus on mutual coherence, which is defined as:

**Definition 8 (Mutual coherence)** *For a matrix $A$, define the mutual coherence as $\mu(A) = \max_{1 \leq i,j \leq n, i \neq j} \frac{|a_i^H a_j|}{\|a_i\|_2 \|a_j\|_2}$.*

By an abuse of notation, let $B$ be the matrix satisfying $b = BX^s$ with $X^s$ being the vectorized version of $X$. We are now ready to state the following theorem:

**Theorem 9 (Recovery using mutual coherence)** *Let $\bar{x}$ be the sparsest solution to* (1). *The solution of CPRL* (5), *$\tilde{X}$, is equal to $\bar{x}\bar{x}^H$ if it has rank 1 and $\|\tilde{X}\|_0 < 0.5(1 + 1/\mu(B))$.*

## 4 Numerical Implementation via ADMM

In addition to the above analysis of guaranteed recovery properties, a critical issue for practitioners is the availability of efficient numerical solvers. Several numerical solvers used in CS may be applied to solve nonsmooth SDPs, which include interior-point methods (*e.g.,* used in CVX [19]), gradient projection methods [4], and augmented Lagrangian methods (ALM) [4]. However, interior-point methods are known to scale badly to moderate-sized convex problems in general. Gradient projection methods also fail to meaningfully accelerate the CPRL implementation due to the complexity of the projection operator. Alternatively, nonsmooth SDPs can be solved by ALM. However, the augmented primal and dual objective functions are still complex SDPs, which are equally expensive to solve in each iteration. In summary, as we will demonstrate in Section 5, CPRL as a nonsmooth complex SDP is categorically more expensive to solve compared to the linear programs underlying CS, and the task exceeds the capability of many popular sparse optimization techniques.

In this paper, we propose a novel solver to the nonsmooth SDP underlying CPRL via the *alternating directions method of multipliers* (ADMM, see for instance [6] and [5, Sec. 3.4]) technique. The motivation to use ADMM are two-fold: 1. It scales well to large data sets. 2. It is known for its fast convergence. There are also a number of strong convergence results [6] which further motivates the choice.

To set the stage for ADMM, rewrite (5) to the equivalent SDP

$$\min_{X_1, X_2, Z} f_1(X_1) + f_2(X_2) + g(Z), \quad \text{subj. to} \quad X_1 - Z = 0, \quad X_2 - Z = 0, \tag{10}$$

where

$$f_1(X) \triangleq \begin{cases} \text{Tr}(X) & \text{if } b_i = Tr(\Phi_i X), i = 1, \ldots, N \\ \infty & \text{otherwise} \end{cases}, \quad f_2(X) \triangleq \begin{cases} 0 & \text{if } X \succeq 0 \\ \infty & \text{otherwise} \end{cases}, \quad g(Z) \triangleq \lambda \|Z\|_1.$$

The update rules of ADMM now lead to the following:

$$\begin{aligned} X_i^{l+1} &= \arg\min_X f_i(X) + \text{Tr}(Y_i^l(X - Z^l)) + \frac{\rho}{2}\|X - Z^l\|_2^2, \\ Z^{l+1} &= \arg\min_Z g(Z) + \sum_{i=1}^2 -\text{Tr}(Y_i^l Z) + \frac{\rho}{2}\|X_i^{l+1} - Z\|_2^2, \\ Y_i^{l+1} &= Y_i^l + \rho(X_i^{l+1} - Z^{l+1}), \end{aligned} \qquad (11)$$

where $X_i, Y_i, Z$ are constrained to stay in the domain of Hermitian matrices. Each of these steps has a tractable calculation. However, the $X_i$, $Y_i$, and $Z$ variables are complex-valued, and, as most of the optimization literature deals with real-valued vectors and symmetric matrices, we will emphasize differences between the real case and complex case. After some simple manipulations, we have:

$$X_1^{l+1} = \text{argmin}_X \|X - (Z^l - \frac{I+Y_1^l}{\rho})\|_2, \quad \text{subj. to} \quad b_i = \text{Tr}(\Phi_i X), \ i = 1, \cdots, N. \qquad (12)$$

Assuming that a feasible solution exists, and defining $\Pi_A$ as the projection onto the convex set given by the linear constraints, the solution is: $X_1^{l+1} = \Pi_A(Z^l - \frac{I+Y_1^l}{\rho})$. This optimization problem has a closed-form solution; converting the matrix optimization problem in (12) into an equivalent vector optimization problem yields a problem of the form: $\min_x \|x - z\|_2$ subj. to $b = Ax$. The answer is given by the pseudo-inverse of $A$, which can be precomputed. This complex-valued problem can be solved by converting the linear constraint in Hermitian matrices into an equivalent constraint on real-valued vectors. This conversion is done by noting that for $n \times n$ Hermitian matrices $A, B$:

$$\begin{aligned} \langle A, B \rangle &= \text{Tr}(AB) = \sum_{i=1}^n \sum_{j=1}^n A_{ij}\overline{B_{ij}} = \sum_{i=1}^n A_{ii}B_{ii} + \sum_{i=1}^n \sum_{j=i+1}^n A_{ij}\overline{B_{ij}} + \overline{A_{ij}}B_{ij} \\ &= \sum_{i=1}^n A_{ii}B_{ii} + \sum_{i=1}^n \sum_{j=i+1}^n 2\,\text{real}(A_{ij})\,\text{real}(B_{ij}) + 2\,\text{imag}(A_{ij})\,\text{imag}(B_{ij}) \end{aligned}$$

So if we define the vector $A^v$ as an $n^2$ vector such that its elements are $A_{ii}$ for $i = 1, \cdots, n$, $\sqrt{2}\,\text{real}(A_{ij})$ for $i = 1, \cdots, n, j = i+1, \cdots, n$, and $\sqrt{2}\,\text{imag}(A_{ij})$ for $i = 1, \cdots, n, j = i+1, \cdots, n$, and similarly define $B^v$, then we can see that $\langle A, B \rangle = \langle A^v, B^v \rangle$. This turns the constraint $b_i = \text{Tr}(\Phi_i X), i = 1, \cdots, N$, into one of the form: $\boldsymbol{b} = [\Phi_1^v \cdots \Phi_N^v]^T X^v$, where each $\Phi_i^v$ is in $\mathbb{R}^{n^2}$. Thus, for this subproblem, the memory usage scales linearly with $N$, the number of measurements, and quadratically with $n$, the dimension of the data. Next, $X_2^{l+1} = \text{argmin}_{X \succeq 0} \|X - (Z^l - \frac{Y_2^l}{\rho})\|_2 = \Pi_{PSD}(Z^l - \frac{Y_2^l}{\rho})$, where $\Pi_{PSD}$ denotes the projection onto the positive-semidefinite cone, which can easily be obtained via eigenvalue decomposition. This holds for real-valued and complex-valued Hermitian matrices. Finally, let $\overline{X}^{l+1} = \frac{1}{2}\sum_{i=1}^2 X_i^{l+1}$ and similarly $\overline{Y}^l$. Then, the $Z$ update rule can be written:

$$Z^{l+1} = \text{argmin}_Z \lambda\|Z\|_1 + \frac{2\rho}{2}\|Z - (\overline{X}^{l+1} + \frac{\overline{Y}^l}{\rho})\|_2^2 = \text{soft}(\overline{X}^{l+1} + \frac{\overline{Y}^l}{\rho}, \frac{\lambda}{2\rho}). \qquad (13)$$

We note that the soft operator in the complex domain must be coded with care. One does not simply check the sign of the difference, as in the real case, but rather the magnitude of the complex number:

$$\text{soft}(x, q) = \begin{cases} 0 & \text{if } |x| \leq q, \\ \frac{|x|-q}{|x|}x & \text{otherwise,} \end{cases} \qquad (14)$$

where $q$ is a positive real number. Setting $l = 0$, the Hermitian matrices $X_i^l, Z_i^l, Y_i^l$ can now be iteratively computed using the ADMM iterations (11). The stopping criterion of the algorithm is given by:

$$\|r^l\|_2 \leq n\epsilon^{abs} + \epsilon^{rel}\max(\|\overline{X}^l\|_2, \|Z^l\|_2), \quad \|s^l\|_2 \leq n\epsilon^{abs} + \epsilon^{rel}\|\overline{Y}^l\|_2, \qquad (15)$$

where $\epsilon^{abs}, \epsilon^{rel}$ are algorithm parameters set to $10^{-3}$ and $r^l$ and $s^l$ are the primal and dual residuals given by: $r^l = (X_1^l - Z^l, X_2^l - Z^l)$, $s^l = -\rho(Z^l - Z^{l-1}, Z^l - Z^{l-1})$. We also update $\rho$ according to the rule discussed in [6]:

$$\rho^{l+1} = \begin{cases} \tau_{incr}\rho^l & \text{if } \|r^l\|_2 > \mu\|s^l\|_2, \\ \rho^l/\tau_{decr} & \text{if } \|s^l\|_2 > \mu\|r^l\|_2, \\ \rho^l & \text{otherwise,} \end{cases} \qquad (16)$$

where $\tau_{incr}, \tau_{decr}$, and $\mu$ are algorithm parameters. Values commonly used are $\mu = 10$ and $\tau_{incr} = \tau_{decr} = 2$.

# 5 Experiment

The experiments in this section are chosen to illustrate the computational performance and scalability of CPRL. Being one of the first papers addressing the CPR problem, existing methods available for comparison are limited. For the CPR problem, to the authors' best knowledge, the only methods developed are the greedy algorithms presented in [25, 27, 28], and GCPRL [26]. The method proposed in [25] handles CPR but is only tailored to random 2D Fourier samples from a 2D array and it is extremely sensitive to initialization. In fact, it would fail to converge in our scenarios of interest. [27] formulates the CPR problem as a nonconvex optimization problem that can be solved by solving a series of convex problems. [28] proposes to alternate between fit the estimate to measurements and thresholding. GCPRL, which stands for greedy CPRL, is a new greedy approximate algorithm tailored to the lifting technique in (5). The algorithm draws inspiration from the matching-pursuit algorithm [22, 1]. In each iteration, the algorithm adds a new nonzero component of $x$ that minimizes the CPRL objective function the most. We have observed that if the number of nonzero elements in $x$ is expected to be low, the algorithm can successfully recover the ground-truth sparse signal while consuming less time compared to interior-point methods for the original SDP.[2] In general, greedy algorithms for solving CPR problems work well when a good guess for the true solution is available, are often computationally efficient but lack theoretical recovery guarantees. We also want to point out that CPRL becomes a special case in a more general framework that extends CS to nonlinear systems (see [1]). In general, nonlinear CS can be solved locally by greedy simplex pursuit algorithms. Its instantiation in PR is the GCPRL algorithm. However, the key benefit of developing the SDP solution for PR in this paper is that the global convergence can be guaranteed.

In this section, we will compare implementations of CPRL using the interior-point method used by CVX [19] and ADMM with the design parameter choice recommended in [6] ($\tau_{incr} = \tau_{decr} = 2$). $\lambda = 10$ will be used in all experiments. We will also compare the results to GCPRL and the PR algorithm PhaseLift [13]. The former is a greedy approximate solution, while the latter does not enforce sparsity and is obtained by setting $\lambda = 0$ in CPRL.

In terms of the scale of the problem, the largest problem we have tested is on a $30 \times 30$ image and is 100-sparse in the Fourier domain with $2400$ measurements. Our experiment is conducted on an IBM x3558 M3 server with two Xeon X5690 processors, 6 cores each at 3.46GHz, 12MB L3 cache, and 96GB of RAM. The execution for recovering one instance takes approximately 36 hours to finish in MATLAB environment, comprising of several tens of thousands of iterations. The average memory usage is 3.5 GB.

## 5.1 A simple simulation

In this example we consider a simple CPR problem to illustrate the differences between CPRL, GCPRL, and PhaseLift. We also compare computational speed for solving the CPR problem and illustrate the theoretical bounds derived in Section 3. Let $x \in \mathbb{C}^{64}$ be a 2-sparse complex signal, $A \triangleq RF$ where $F \in \mathbb{C}^{64 \times 64}$ is the Fourier transform matrix and $R \in \mathbb{C}^{32 \times 64}$ a random projection matrix (generated by sampling a unit complex Gaussian), and let the measurements $b$ satisfy the PR relation (1). The left plot of Figure 1 gives the recovered signal $x$ using CPRL, GCPRL and PhaseLift. As seen, CPRL and GCPRL correctly identify the two nonzero elements in $x$ while PhaseLift fails to identify the true signal and gives a dense estimate. These results are rather typical (see the MCMC simulation in [26]). For very sparse examples, like this one, CPRL and GCPRL often both succeed in finding the ground truth (even though we have twice as many unknowns as measurements). PhaseLift, on the other side, does not favor sparse solutions and would need considerably more measurements to recover the 2-sparse signal. The middle plot of Figure 1 shows the computational time needed to solve the nonsmooth SDP of CPRL using CVX, ADMM, and GCPRL. It shows that ADMM is the fastest and that GCPRL outperforms CVX. The right plot of Figure 1 shows the mutual coherence bound $0.5(1 + 1/\mu(B))$ for a number of different $N$'s and $n$'s, $A \triangleq RF$, $F \in \mathbb{C}^{n \times n}$ the Fourier transform matrix and $R \in \mathbb{C}^{N \times n}$ a random projection matrix. This is of interest since Theorem 9 states that when the CRPL solution $\tilde{X}$ satisfies $\|\tilde{X}\|_0 < 0.5(1 + 1/\mu(B))$ and has rank 1, then $\tilde{X} = \bar{x}\bar{x}^H$, where $\bar{x}$ is the sparsest solution to (1). From

the plot it can be concluded that if the CPRL solution $\tilde{X}$ has rank 1 and only a single nonzero component for a choice of $125 \geq n$, $N \geq 5$, Theorem 9 guarantees that $\tilde{X} = \bar{x}\bar{x}^H$. We also observe that Theorem 9 is conservative, since we previously saw that 2 nonzero components could be recovered correctly for $n = 64$ and $N = 32$. In fact, numerical simulation can be used to show that $N = 30$ suffices to recover the ground truth in 95 out of 100 runs [26].

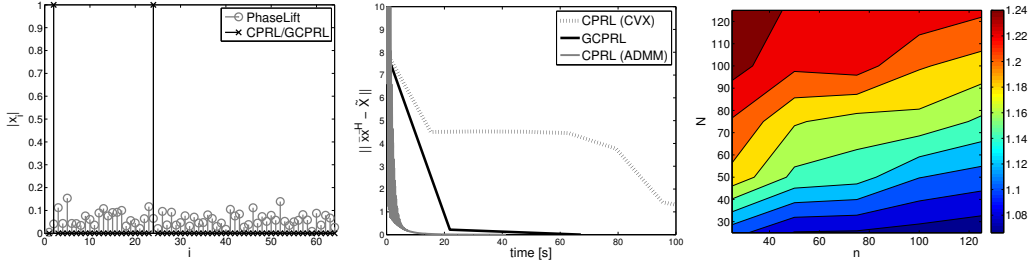

Figure 1: **Left:** The magnitude of the estimated signal provided by CPRL, GCPRL and PhaseLift. **Middle:** The residual $\|\bar{x}\bar{x}^H - \tilde{X}\|_2$ plotted against time for ADMM (gray line), GCPRL (solid black line) and CVX (dashed black line). **Right:** A contour plot of the quantity $0.5(1 + 1/\mu(B))$. $\mu$ is taken as the average over 10 realizations of the data.

## 5.2 Compressive sampling and PR

One of the motivations of presented work and CPRL is that it enables compressive sensing for PR problems. To illustrate this, consider the $20 \times 20$ complex image in Figure 2 Left. To measure the image, we could measure each pixel one-by-one. This would require us to sample 400 times. What CS proposes is to measure linear combinations of samples rather than individual pixels. It has been shown that the original image can be recovered from far fewer samples than the total number of pixels in the image. The gain using CS is hence that fewer samples are needed. However, traditional CS only discuss linear relations between measurements and unknowns.

To extend CS to PR applications, consider again the complex image in Figure 2 Left and assume that we only can measure intensities or intensities of linear combinations of pixels. Let $R \in \mathbb{C}^{N \times 400}$ capture how intensity measurements $b$ are formed from linear combinations of pixels in the image, $b = |Rz|^2$ ($z$ is a vectorized version of the image). An essential part in CS is also to find a dictionary (possibly overcomplete) in which the image can be represented using only a few basis images. For classical CS applications, dictionaries have been derived. For applying CS to the PR applications, dictionaries are needed and a topic for future research. We will use a 2D inverse Fourier transform dictionary in our example and arrange the basis vectors as columns in $F \in \mathbb{C}^{400 \times 400}$.

If we choose $N = 400$ and generate $R$ by sampling from a unit Gaussian distribution and set $A = RF$, CPRL recovers exactly the true image. This is rather remarkable since the PR relation (1) is nonlinear in the unknown $x$ and $N \gg n$ measurements are in general needed for a unique solution. If we instead sample the intensity of each pixel, one-by-one, neither CPRL or PhaseLift recover the true image. If we set $A = R$ and do not care about finding a dictionary, we can use a classical PR algorithm to recover the true image. If PhaseLift is used, $N = 1600$ measurements are sufficient to recover the true image. The main reasons for the low number of samples needed in CPRL is that we managed to find a good dictionary (20 basis images were needed to recover the true image) and CPRL's ability to recover the sparsest solution. In fact, setting $A = RF$, PhaseLift still needs 1600 measurements to recover the true solution.

## 5.3 The Shepp-Logan phantom

In this last example, we again consider the recovery of complex valued images from random samples. The motivation is twofold: Firstly, it illustrates the scalability of the ADMM implementation. In fact, ADMM has to be used in this experiment as CVX cannot handle the CPRL problem in this scale. Secondly, it illustrates that CPRL can provide approximate solutions that are visually close to the ground-truth images. Consider now the image in Figure 2 Middle Left. This $30 \times 30$ Shepp-Logan phantom has a 2D Fourier transform with 100 nonzero coefficients. We generate $N$ linear combinations of pixels as in the previous example and square the measurements, and then apply

CPRL and PhaseLift with a 2D Fourier dictionary. The middel image in Figure 2 shows the recovered result using PhaseLift with $N = 2400$, the second image from the right shows the recovered result using CPRL with the same number $N = 2400$ and the right image is the recovered result using CPRL with $N = 1500$. The number of measurements with respect to the sparsity in $x$ is too low for both CPRL and PhaseLift to perfectly recover $z$. However, CPRL provides a much better approximation and outperforms PhaseLift visually even though it uses considerably fewer measurements.

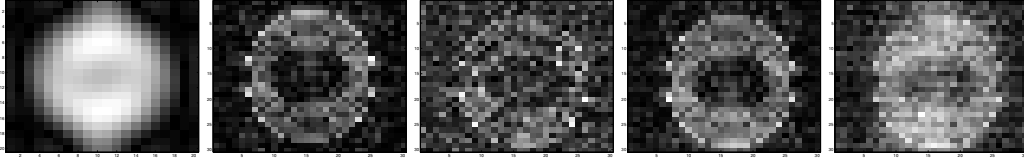

Figure 2: **Left:** Absolute value of the 2D inverse Fourier transform of $x$, $|Fx|$, used in the experiment in Section 5.2. **Middle Left:** Ground truth for the experiment in Section 5.3. **Middle:** Recovered result using PhaseLift with $N = 2400$. **Middle Right:** CPRL with $N = 2400$. **Right:** CPRL with $N = 1500$.

## 6   Future Directions

The SDP underlying CPRL scales badly with the number of unknowns or basis vectors in the dictionary. Therefore, learning a suitable dictionary for a specific application becomes even more critical than that in traditional linear CS setting. We also want to point out that when classical CS was first studied, many of today's accelerated numerical algorithms were not available. We are very excited about the new problem to improve the speed of SDP algorithms in sparse optimization, and hope our paper would foster the community's interest to address this challenge collaboratively. One interesting direction might be to use ADMM to solve the dual of (5), see for instance [30, 31]. Another possible direction is the outer approximation methods [21].

## 7   Acknowledgement

Ohlsson is partially supported by the Swedish foundation for strategic research in the center MOVIII, the Swedish Research Council in the Linnaeus center CADICS, the European Research Council under the advanced grant LEARN, contract 267381, and a postdoctoral grant from the Sweden-America Foundation, donated by ASEA's Fellowship Fund, and by a postdoctoral grant from the Swedish Research Council. Yang is supported by ARO 63092-MA-II. Dong is supported by the NSF Graduate Research Fellowship under grant DGE 1106400, and by the Team for Research in Ubiquitous Secure Technology (TRUST), which receives support from NSF (award number CCF-0424422). The authors also want to acknowledge useful input from Stephen Boyd and Yonina Eldar.

## Footnotes

[1]In this paper, $\|X\|_1$ for a matrix $X$ denotes the entry-wise $\ell_1$-norm, and $\|X\|_2$ denotes the Frobenius norm.

[2] We have also tested an off-the-shelf toolbox that solves convex cone problems, called TFOCS [2]. Unfortunately, TFOCS cannot be applied directly to solving the nonsmooth SDP in CPRL.

## References

[1] A. Beck and Y. C. Eldar. Sparsity constrained nonlinear optimization: Optimality conditions and algorithms. Technical Report arXiv:1203.4580, 2012.

[2] S. Becker, E. Candès, and M. Grant. Templates for convex cone problems with applications to sparse signal recovery. *Mathematical Programming Computation*, 3(3), 2011.

[3] R. Berinde, A. Gilbert, P. Indyk, H. Karloff, and M. Strauss. Combining geometry and combinatorics: A unified approach to sparse signal recovery. In *Communication, Control, and Computing, 2008 46th Annual Allerton Conference on*, pages 798–805, September 2008.

[4] D. P. Bertsekas. *Nonlinear Programming*. Athena Scientific, 1999.

[5] D. P. Bertsekas and J. N. Tsitsiklis. *Parallel and Distributed Computation: Numerical Methods*. Athena Scientific, 1997.

[6] S. Boyd, N. Parikh, E. Chu, B. Peleato, and J. Eckstein. Distributed optimization and statistical learning via the alternating direction method of multipliers. *Foundations and Trends in Machine Learning*, 2011.

[7] A. Bruckstein, D. Donoho, and M. Elad. From sparse solutions of systems of equations to sparse modeling of signals and images. *SIAM Review*, 51(1):34–81, 2009.

[8] E. Candès. Compressive sampling. In *Proceedings of the International Congress of Mathematicians*, volume 3, pages 1433–1452, Madrid, Spain, 2006.

[9] E. Candès. The restricted isometry property and its implications for compressed sensing. *Comptes Rendus Mathematique*, 346(9–10):589–592, 2008.

[10] E. Candès, Y. Eldar, T. Strohmer, and V. Voroninski. Phase retrieval via matrix completion. Technical Report arXiv:1109.0573, Stanford University, September 2011.

[11] E. Candès, X. Li, Y. Ma, and J. Wright. Robust Principal Component Analysis? *Journal of the ACM*, 58(3), 2011.

[12] E. Candès, J. Romberg, and T. Tao. Robust uncertainty principles: Exact signal reconstruction from highly incomplete frequency information. *IEEE Transactions on Information Theory*, 52:489–509, February 2006.

[13] E. Candès, T. Strohmer, and V. Voroninski. PhaseLift: Exact and stable signal recovery from magnitude measurements via convex programming. Technical Report arXiv:1109.4499, Stanford University, September 2011.

[14] S. Chen, D. Donoho, and M. Saunders. Atomic decomposition by basis pursuit. *SIAM Journal on Scientific Computing*, 20(1):33–61, 1998.

[15] A. d'Aspremont, L. El Ghaoui, M. Jordan, and G. Lanckriet. A direct formulation for Sparse PCA using semidefinite programming. *SIAM Review*, 49(3):434–448, 2007.

[16] D. Donoho. Compressed sensing. *IEEE Transactions on Information Theory*, 52(4):1289–1306, April 2006.

[17] D. Donoho and M. Elad. Optimally sparse representation in general (nonorthogonal) dictionaries via $\ell_1$-minimization. *PNAS*, 100(5):2197–2202, March 2003.

[18] J. Fienup. Reconstruction of a complex-valued object from the modulus of its Fourier transform using a support constraint. *Journal of Optical Society of America A*, 4(1):118–123, 1987.

[19] M. Grant and S. Boyd. CVX: Matlab software for disciplined convex programming, version 1.21. `http://cvxr.com/cvx`, August 2010.

[20] D. Kohler and L. Mandel. Source reconstruction from the modulus of the correlation function: a practical approach to the phase problem of optical coherence theory. *Journal of the Optical Society of America*, 63(2):126–134, 1973.

[21] H. Konno, J. Gotoh, T. Uno, and A. Yuki. A cutting plane algorithm for semi-definite programming problems with applications to failure discriminant analysis. *Journal of Computational and Applied Mathematics*, 146(1):141–154, 2002.

[22] S. Mallat and Z. Zhang. Matching pursuits with time-frequency dictionaries. *IEEE Transactions on Signal Processing*, 41(12):3397–3415, December 1993.

[23] S. Marchesini. Phase retrieval and saddle-point optimization. *Journal of the Optical Society of America A*, 24(10):3289–3296, 2007.

[24] R. Millane. Phase retrieval in crystallography and optics. *Journal of the Optical Society of America A*, 7:394–411, 1990.

[25] M. Moravec, J. Romberg, and R. Baraniuk. Compressive phase retrieval. In *SPIE International Symposium on Optical Science and Technology*, 2007.

[26] H. Ohlsson, A. Y. Yang, R. Dong, and S. Sastry. Compressive Phase Retrieval From Squared Output Measurements Via Semidefinite Programming. Technical Report arXiv:1111.6323, University of California, Berkeley, November 2011.

[27] Y. Shechtman, Y. C. Eldar, A. Szameit, and M. Segev. Sparsity based sub-wavelength imaging with partially incoherent light via quadratic compressed sensing. *Opt. Express*, 19(16):14807–14822, Aug 2011.

[28] A. Szameit, Y. Shechtman, E. Osherovich, E. Bullkich, P. Sidorenko, H. Dana, S. Steiner, E. B. Kley, S. Gazit, T. Cohen-Hyams, S. Shoham, M. Zibulevsky, I. Yavneh, Y. C. Eldar, O. Cohen, and M. Segev. Sparsity-based single-shot subwavelength coherent diffractive imaging. *Nature Materials*, 11(5):455–459, May 2012.

[29] A. Walther. The question of phase retrieval in optics. *Optica Acta*, 10:41–49, 1963.

[30] Z. Wen, D. Goldfarb, and W. Yin. Alternating direction augmented lagrangian methods for semidefinite programming. *Mathematical Programming Computation*, 2:203–230, 2010.

[31] Z. Wen, C. Yang, X. Liu, and S. Marchesini. Alternating direction methods for classical and ptychographic phase retrieval. *Inverse Problems*, 28(11):115010, 2012.

